# Brain Inspired Reinforcement Learning

**François Rivest**[*]          **Yoshua Bengio**
Département d'informatique et de recherche opérationnelle
Université de Montréal
CP 6128 succ. Centre Ville, Montréal, QC H3C 3J7, Canada
*francois.rivest@mail.mcgill.ca          bengioy@iro.umontreal.ca*

**John Kalaska**
Département de physiologie
Université de Montréal
*kalaskaj@physio.umontreal.ca*

## Abstract

Successful application of reinforcement learning algorithms often involves considerable hand-crafting of the necessary non-linear features to reduce the complexity of the value functions and hence to promote convergence of the algorithm. In contrast, the human brain readily and autonomously finds the complex features when provided with sufficient training. Recent work in machine learning and neurophysiology has demonstrated the role of the basal ganglia and the frontal cortex in mammalian reinforcement learning. This paper develops and explores new reinforcement learning algorithms inspired by neurological evidence that provides potential new approaches to the feature construction problem. The algorithms are compared and evaluated on the Acrobot task.

## 1   Introduction

Reinforcement learning algorithms often face the problem of finding useful complex non-linear features [1]. Reinforcement learning with non-linear function approximators like backpropagation networks attempt to address this problem, but in many cases have been demonstrated to be non-convergent [2]. The major challenge faced by these algorithms is that they must learn a value function instead of learning the policy, motivating an interest in algorithms directly modifying the policy [3].

In parallel, recent work in neurophysiology shows that the basal ganglia can be modeled by an actor-critic version of temporal difference (TD) learning [4][5][6], a well-known reinforcement learning algorithm. However, the basal ganglia do not, by themselves, solve the problem of finding complex features. But the frontal cortex, which is known to play an important role in planning and decision-making, is tightly linked with the basal ganglia. The nature or their interaction is still poorly understood, and is generating a growing interest in neurophysiology.

---

[*] URL: http://www.iro.umontreal.ca/~rivestfr

This paper presents new algorithms based on current neurophysiological evidence about brain functional organization. It tries to devise biologically plausible algorithms that may help overcome existing difficulties in machine reinforcement learning. The algorithms are tested and compared on the Acrobot task. They are also compared to TD using standard backpropagation as function approximator.

## 2  Biological Background

The mammalian brain has multiple learning subsystems. Major learning components include the neocortex, the hippocampal formation (explicit memory storage system), the cerebellum (adaptive control system) and the basal ganglia (reinforcement learning, also known as instrumental conditioning).

The cortex can be argued to be equipotent, meaning that, given the same input, any region can learn to perform the same computation. Nevertheless, the frontal lobe differs by receiving a particularly prominent innervation of a specific type of neurotransmitter, namely dopamine. The large frontal lobe in primates, and especially in humans, distinguishes them from lower mammals. Other regions of the cortex have been modeled using unsupervised learning methods such as ICA [7], but models of learning in the frontal cortex are only beginning to emerge.

The frontal dopaminergic input arises in a part of the basal ganglia called ventral tegmental area (VTA) and the substantia nigra (SN). The signal generated by dopaminergic (DA) neurons resembles the effective reinforcement signal of temporal difference (TD) learning algorithms [5][8]. Another important part of the basal ganglia is the striatum. This structure is made of two parts, the matriosome and the striosome. Both receive input from the cortex (mostly frontal) and from the DA neurons, but the striosome projects principally to DA neurons in VTA and SN. The striosome is hypothesized to act as a reward predictor, allowing the DA signal to compute the difference between the expected and received reward. The matriosome projects back to the frontal lobe (for example, to the motor cortex). Its hypothesized role is therefore in action selection [4][5][6].

Although there have been several attempts to model the interactions between the frontal cortex and basal ganglia, little work has been done on learning in the frontal cortex. In [9], an adaptive learning system based on the cerebellum and the basal ganglia is proposed. In [10], a reinforcement learning model of the hippocampus is presented. In this paper, we do not attempt to model neurophysiological data per se, but rather to develop, from current neurophysiological knowledge, new and efficient biologically plausible reinforcement learning algorithms.

## 3  The Model

All models developed here follow the architecture depicted in Figure 1. The first layer (I) is the input layer, where activation represents the current state. The second layer, the hidden layer (H), is responsible for finding the non-linear features necessary to solve the task. Learning in this layer will vary from model to model. Both the input and the hidden layer feed the parallel actor-critic layers (A and V) which are the computational analogs of the striatal matriosome and striosome, respectively. They represent a linear actor-critic implementation of TD.

The neurological literature reports an uplink from V and the reward to DA neurons which sends back the effective reinforcement signal $e$ (dashed lines) to A, V and H. The A action units usually feed into the motor cortex, which controls muscle activation. Here, A's are considered to represent the possible actions. The basal ganglia receive input mainly from the frontal cortex and the dopaminergic signal

(*e*). They also receive some input from parietal cortex (which, as opposed to the frontal cortex, does not receive DA input, and hence, may be unsupervised). H will represent frontal cortex when given *e* and non-frontal cortex when not. The weights *W, v* and *U* correspond to weights into the layers A, V and H respectively (*e* is not weighted).

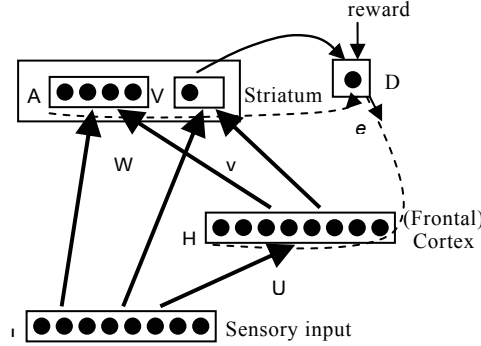

Figure 1: Architecture of the models.

Let $x_t$ be the vector of the input layer activations based on the state of the environment at time *t*. Let *f* be the sigmoidal activation function of hidden units in H. Then $y_t = [f(u_1 x_t), \ldots f(u_n x_t)]^T$, the vector of activations of the hidden layer at time *t*, and where $u_i$ is a row of the weight matrix *U*. Let $z_t = [x_t^T \ y_t^T]^T$ be the state description formed by the layers I and H at time *t*.

### 3.1   Actor-critic

The actor-critic model of the basal ganglia developed here is derived from [4]. It is very similar to the basal ganglia model in [5] which has been used to simulate neurophysiological data recorded while monkeys were learning a task [6]. All units are linear weighted sums of activity from the previous layers. The actor units behave under a winner-take-all rule. The winner's activity settles to 1, and the others to 0. The initial weights are all equal and non-negative in order to obtain an initial optimist policy. Beginning with an overestimate of the expected reward leads every action to be negatively corrected, one after the other until the best one remains. This usually favors exploration.

Then $V(z_t) = v^T z_t$. Let $b_t = W z_t$ be the vector of activation of the actor layer before the winner take all processing. Let $a_t = argmax(b_{t,i})$ be the winning action index at time *t*, and let the vector $c_t$ be the activation of the layer A after the winner take all processing such that $c_{t,a} = 1$ if $a = a_t$, 0 otherwise.

### 3.1.1   Formal description

TD learns a function V of the state that should converge to the expected total discounted reward. In order to do so, it updates V such that

$$V(z_{t-1}) \rightarrow E[r_t + \gamma V(z_t)]$$

where $r_t$ is the reward at time *t* and $\gamma$ the discount factor. A simple way to achieve that is to transform the problem into an optimization problem where the goal is to minimize:

$$E = [V(z_{t-1}) - r_t - \gamma V(z_t)]^2$$

It is also useful at this point, to introduce the TD effective reinforcement signal, equivalent to the dopaminergic signal [5]:

$$e_t = r_t + \gamma V(z_t) - V(z_{t-1})$$

Thus: $E = e_t{}^2$.

A learning rule for the weights $v$ of V can then be devised by finding the gradient of E with respect to the weights $v$. Here, V is the weighted sum of the activity of I and H. Thus, the gradient is given by

$$\frac{\partial E}{\partial v} = 2e_t \left[\gamma z_t - z_{t-1}\right]$$

Adding a learning rate and negating the gradient for minimization gives the update:

$$\Delta v = \alpha e_t \left[z_{t-1} - \gamma z_t\right]$$

Developing a learning rule for the actor units and their weights $W$ using a cost function is a bit more complex. One approach is to use the tri-hebbian rule

$$\Delta W = \alpha e_t c_{t-1} z_{t-1}{}^T$$

Remark that only the row vector of weights of the winning action is modified.

This rule was first introduced, but not simulated, in [4]. It associates the error $e$ to the last selected action. If the reward is higher than expected ($e > 0$), than the action units activated by the previous state should be reinforced. Conversely, if it is less than expected ($e < 0$), than the winning actor unit activity should be reduced for that state. This is exactly what this tri-hebbian rule does.

### 3.1.2 Biological justification

[4] presented the first description of an actor-critic architecture based on data from the basal ganglia that resemble the one here. The major difference is that the V update rule did not use the complete gradient information.

A similar version was also developed in [5], but with little mathematical justification for the update rule. The model presented here is simpler and the critic update rule is basically the same, but justified neurologically. Our model also has a more realistic actor update rule consistent with neurological knowledge of plasticity in the corticostriatal synapses [11] (H to V weights). The main purpose of the model presented in [5] was to simulate dopaminergic activity for which V is the most important factor, and in this respect, it was very successful [6].

### 3.2 Hidden Layer

Because the reinforcement learning layer is linear, the hidden layer must learn the necessary non-linearity to solve the task. The rules below are attempts at neurologically plausible learning rules for the cortex, assuming it has no clear supervision signal other than the DA signal for the frontal cortex. All hidden units weight vectors are initialized randomly and scaled to norm 1 after each update.

- **Fixed random**

This is the baseline model to which the other algorithms will be compared. The hidden layer is composed of randomly generated hidden units that are not trained.

- **ICA**

In [7], the visual cortex was modeled by an ICA learning rule. If the non-frontal cortex is equipotent, then any region of the cortex could be successfully modeled using such a generic rule. The idea of combining unsupervised learning with reinforcement learning has already proven useful [1], but the unsupervised features were trained prior to the reinforcement training. On the other hand, [12] has shown that different systems of this sort could learn concurrently. Here, the ICA rule from [13] will be used as the hidden layer. This means that the hidden units are learning to reproduce the independent source signals at the origin of the observed mixed signal.

- **Adaptive ICA (*e*-ICA)**

If H represents the frontal cortex, then an interesting variation of ICA is to multiply its update term by the DA signal *e*. The size of *e* may act as an adaptive learning rate whose source is the reinforcement learning system critic. Also, if the reward is less than expected ($e < 0$), the features learned by the ICA unit may be more counterproductive than helpful, and *e* pushes the learning away from those features.

- *e*-**gradient method**

Another possible approach is to base the update rule on the derivative of the objective function E applied to the hidden layer weights *U*, but constraining the update rule to only use information available locally. Let *f'* be the derivative of *f*, then the gradient of E with respect to *U* is approximated by:

$$\frac{\partial E}{\partial u_i} = 2e_t \left[ v_i f'(u_i x_t) x_t - v_i f'(u_i x_{t-1}) x_{t-1} \right]$$

Negating the gradient for minimization, adding a learning rate and removing the non-local weight information, gives the weight update rule:

$$\Delta u_i = \alpha e_t \left[ f'(u_i x_{t-1}) x_{t-1} - \gamma f'(u_i x_t) x_t \right]$$

Using the value of the weights *v* would lead to a rule that use non-local information. The cortex is unlikely to have this and might consider all the weights in *v* to be equal to some constant.

To avoid neurons all moving in the same direction uniformly, we encourage the units on the hidden layer to minimize their covariance. This can be achieved by adding an inhibitory neuron. Let $q_t$ be the average activity of the hidden units at time *t*, i.e., the inhibitory neuron activity. Let $\bar{q}_t$ be the moving exponential average of $q_t$. Since

$$Var[q_t] = \frac{1}{n^2} \sum_{i,j} \text{cov}(y_{t,i}, y_{t,j}) \cong TimeAverage\left( (q_t - \bar{q}_t)^2 \right)$$

and ignoring the *f*'s non-linearity , the gradient of the *Var[qt]* with respect to the weights *U* is approximated by:

$$\frac{\partial Var[q_t]}{\partial u_i} = 2(q_t - \bar{q}_t) x_t$$

Combined with the previous equation, this results in a new update rule:

$$\Delta u_i = \alpha e_t \left[ f'(u_i x_{t-1}) x_{t-1} - \gamma f'(u_i x_t) x_t \right] + \alpha \left[ \bar{q}_t - q_t \right] x_t$$

When allowing the discount factor to be different on the hidden layer, we found that $\gamma = 0$ gave much better results (*e*-gradient(0)).

## 4   Simulations & Results

All models of section 3 were run on the Acrobot task [8]. This task consists of a two-link pendulum with torque on the middle joint. The goal is to bring the tip of the second pole in a totally upright position.

### 4.1   The task: Acrobot

The input was coded using 12 equidistant radial basis functions for each angle and 13 equidistant radial basis functions for each angular velocity, for a total of 50 non-negative inputs. This somewhat simulates the input from joint-angle receptors. A reward of 1 was given only when the final state was reached (in all other case, the reward of an action was 0). Only 3 actions were available (3 actor units), either -1, 0 or 1 unit of torque. The details can be found in [8].

50 networks with different random initialization where run for all models for 100 episodes (an episode is the sequence of steps the network performs to achieve the goal from the start position). Episodes were limited to 10000 steps. A number of learning rate values were tried for each model (actor-critic layer learning rate, and hidden layer learning rate). The selected parameters were the ones for which the average number of steps per episode plus its standard deviation was the lowest. All hidden layer models got a learning rate of 0.1.

### 4.2   Results

Figure 2 displays the learning curves of every model evaluated. Three variables were compared: overall learning performance (in number of steps to success per episode), final performance (number of steps on the last episode), and early learning performance (number of steps for the first episode).

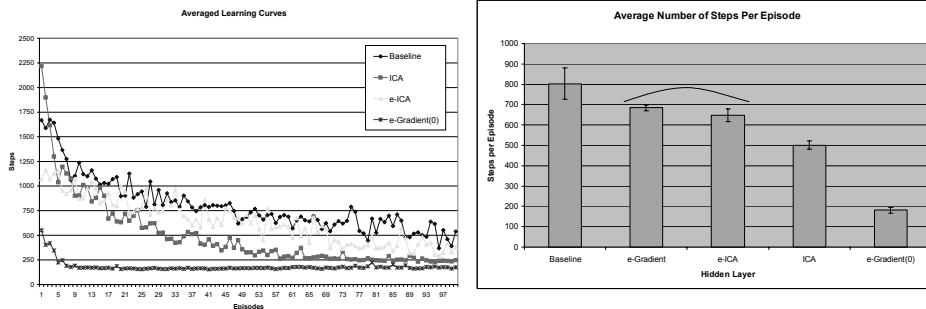

Figure 2: Learning curves of the models.

Figure 3: Average number of steps per episode with 95% confidence interval.

### 4.2.1   Space under the learning curve

Figure 3 shows the average steps per episode for each model in decreasing order. All models needed fewer steps on average than baseline (which has no training at the hidden layer). In order to assess the performance of the models, an ANOVA analysis of the average number of steps per episode over the 100 episodes was performed. Scheffé post-hoc analysis revealed that the performance of every model

was significantly different from every other, except for *e*-gradient and *e*-ICA (which are not significantly different from each other).

### 4.2.2 Final performance

ANOVA analysis was also used to determine the final performance of the models, by comparing the number of steps on the last episode. Scheffé test results showed that all but *e*-ICA are significantly better than the baseline. Figure 4 shows the results on the last episode in increasing order. The curved lines on top show the homogeneous subsets.

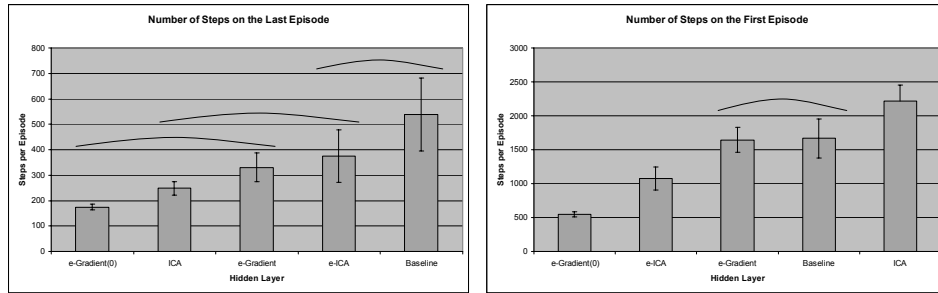

Figure 4: Number of steps on the last episode with 95% confidence interval.

Figure 5: Number of steps on the first episode with 95% confidence interval.

### 4.2.3 Early learning

Figure 2 shows that the models also differed in their initial learning. To assess how different those curves are, an ANOVA was run on the number of steps on the very first episode. Under this measure, *e*-gradient(0) and *e*-ICA were significantly faster than the baseline and ICA was significantly slower (Figure 5).

It makes sense for ICA to be slower at the beginning, since it first has to stabilize for the RL system to be able to learn from its input. Until the ICA has stabilized, the RL system has moving inputs, and hence cannot learn effectively. Interestingly, *e*-ICA was protected against this effect, having a start-up significantly faster than the baseline. This implies that the *e* signal could control the ICA learning to move synergistically with the reinforcement learning system.

### 4.3 External comparison

Acrobot was also run using standard backpropagation with TD and $\varepsilon$-Greedy policy. In this setup, a neural network of 50 inputs, 50 hidden sigmoidal units, and 1 linear output was used as function approximator for V. The network had cross-connections and its weights were initialized as in section 3 such that both architectures closely matched in terms of power. In this method, the RHS of the TD equation is used as a constant target value for the LHS. A single gradient was applied to minimize the squared error after the result of each action. Although not different from the baseline on the first episode, it was significantly worst on overall and final performance, unable to constantly improve. This is a common problem when using backprop networks in RL without handcrafting the necessary complex features. We also tried SARSA (using one network per action), but results were worst than TD.

The best result we found in the literature on the exact same task are from [8]. They used SARSA($\lambda$) with a linear combination of tiles. Tile coding discretized the input space into small hyper-cubes and few overlapping tilings were used. From available reports, their first trial could be slower than *e*-gradient(0) but they could reach better

final performance after more than 100 episodes with a final average of 75 steps (after 500 episodes). On the other hand, their function had about 75000 weights while all our models used 2900 weights.

## 5   Discussion

In this paper we explored a new family of biologically plausible reinforcement learning algorithms inspired by models of the basal ganglia and the cortex. They use a linear actor-critic model of the basal ganglia and were extended with a variety of unsupervised and partially supervised learning algorithms inspired by brain structures. The results showed that pure unsupervised learning was slowing down learning and that a simple quasi-local rule at the hidden layer greatly improved performance. Results also demonstrated the advantage of such a simple system over the use of function approximators such as backpropagation. Empirical results indicate a strong potential for some of the combinations presented here. It remains to test them on further tasks, and to compare them to more reinforcement learning algorithms. Possible loops from the actor units to the hidden layer are also to be considered.

### Acknowledgments

This research was supported by a New Emerging Team grant to John Kalaska and Yoshua Bengio from the CIHR. We thank Doina Precup for helpful discussions.

### References

[1] Foster, D. & Dayan, P. (2002) Structure in the space of value functions. *Machine Learning* **49**(2):325-346.

[2] Tsitsiklis, J.N. & Van Roy, B. (1996) Featured-based methods for large scale dynamic programming. *Machine Learning* **22**:59-94.

[3] Sutton, R.S., McAllester, D., Singh, S. & Mansour, Y. (2000) Policy gradient methods for reinforcement learning with function approximation. *Advances in Neural Information Processing Systems 12*, pp. 1057-1063. MIT Press.

[4] Barto A.G. (1995) Adaptive critics and the basal ganglia. In *Models of Information Processing in the Basal Ganglia*, pp.215-232. Cambridge, MA: MIT Press.

[5] Suri, R.E. & Schultz, W. (1999) A neural network model with dopamine-like reinforcement signal that learns a spatial delayed response task. *Neuroscience* **91**(3):871-890.

[6] Suri, R.E. & Schultz, W. (2001) Temporal difference model reproduces anticipatory neural activity. *Neural Computation* **13**:841-862.

[7] Doi, E., Inui, T., Lee, T.-W., Wachtler, T. & Sejnowski, T.J. (2003) Spatiochromatic receptive field properties derived from information-theoritic analysis of cone mosaic responses to natural scenes. *Neural Computation* **15**:397-417.

[8] Sutton R.S. & Barto A.G. (1998) *Reinforcement Learning: An Introduction*. Cambridge, MA: MIT Press.

[9] Doya K. (1999) What are the computations of the cerebellum, the basal ganglia and the cerebral cortex? *Neural Networks* **12**:961-974.

[10] Foster, D.J., Morris, R.G.M., & Dayan, P. (2000) A model of hippocampally dependent navigation, using the temporal difference learning rule. *Hippocampus* **10**:1-16.

[11] Wickens, J. & Kötter, R. (1995) Cellular models of reinforcement. In *Models of Information Processing in the Basal Ganglia*, pp.187-214. Cambridge, MA: MIT Press.

[12] Whiteson, S. & Stone, P. (2003) Concurrent layered learning. In *Proceedings of the 2nd Internaltional Joint Conference on Autonomous Agents & Multi-agent Systems*.

[13] Amari, S-I (1999) Natural gradient learning for over- and under-complete bases in ICA. *Neural Computation* **11**:1875-1883.
